# On-line Reinforcement Learning Using Incremental Kernel-Based Stochastic Factorization

**André M. S. Barreto**
School of Computer Science
McGill University
Montreal, Canada
amsb@cs.mcgill.ca

**Doina Precup**
School of Computer Science
McGill University
Montreal, Canada
dprecup@cs.mcgill.ca

**Joelle Pineau**
School of Computer Science
McGill University
Montreal, Canada
jpineau@cs.mcgill.ca

## Abstract

Kernel-based stochastic factorization (KBSF) is an algorithm for solving reinforcement learning tasks with continuous state spaces which builds a Markov decision process (MDP) based on a set of sample transitions. What sets KBSF apart from other kernel-based approaches is the fact that the size of its MDP is independent of the number of transitions, which makes it possible to control the trade-off between the quality of the resulting approximation and the associated computational cost. However, KBSF's memory usage grows linearly with the number of transitions, precluding its application in scenarios where a large amount of data must be processed. In this paper we show that it is possible to construct KBSF's MDP in a fully incremental way, thus freeing the space complexity of this algorithm from its dependence on the number of sample transitions. The incremental version of KBSF is able to process an arbitrary amount of data, which results in a model-based reinforcement learning algorithm that can be used to solve continuous MDPs in both off-line and on-line regimes. We present theoretical results showing that KBSF can approximate the value function that would be computed by conventional kernel-based learning with arbitrary precision. We empirically demonstrate the effectiveness of the proposed algorithm in the challenging three-pole balancing task, in which the ability to process a large number of transitions is crucial for success.

## 1 Introduction

The task of learning a policy for a sequential decision problem with continuous state space is a long-standing challenge that has attracted the attention of the reinforcement learning community for years. Among the many approaches that have been proposed to solve this problem, *kernel-based reinforcement learning* (KBRL) stands out for its good theoretical guarantees [1, 2]. KBRL solves a continuous state-space Markov decision process (MDP) using a finite model constructed based on sample transitions only. By casting the problem as a non-parametric approximation, it provides a statistically consistent way of approximating an MDP's value function. Moreover, since it comes down to the solution of a finite model, KBRL always converges to a unique solution.

Unfortunately, the good theoretical properties of kernel-based learning come at a price: since the model constructed by KBRL grows with the amount of sample transitions, the number of operations performed by this algorithm quickly becomes prohibitively large as more data become available. Such a computational burden severely limits the applicability of KBRL to real reinforcement learning (RL) problems. Realizing that, many researchers have proposed ways of turning KBRL into a more practical tool [3, 4, 5]. In this paper we focus on our own approach to leverage KBRL, an algorithm called *kernel-based stochastic factorization* (KBSF) [4].

KBSF uses KBRL's kernel-based strategy to perform a soft aggregation of the states of its MDP. By doing so, our algorithm is able to summarize the information contained in KBRL's model in an MDP whose size is independent of the number of sample transitions. KBSF enjoys good theoretical

guarantees and has shown excellent performance on several tasks [4]. The main limitation of the algorithm is the fact that, in order to construct its model, it uses an amount of memory that grows linearly with the number of sample transitions. Although this is a significant improvement over KBRL, it still hinders the application of KBSF in scenarios in which a large amount of data must be processed, such as in complex domains or in on-line reinforcement learning.

In this paper we show that it is possible to construct KBSF's MDP in a fully incremental way, thus freeing the space complexity of this algorithm from its dependence on the number of sample transitions. In order to distinguish it from its original, batch counterpart, we call this new version of our algorithm *incremental KBSF*, or *i*KBSF for short. As will be seen, *i*KBSF is able to process an arbitrary number of sample transitions. This results in a model-based RL algorithm that can be used to solve continuous MDPs in both off-line and on-line regimes.

A second important contribution of this paper is a theoretical analysis showing that it is possible to control the error in the value-function approximation performed by KBSF. In our previous experiments with KBSF, we defined the model used by this algorithm by clustering the sample transitions and then using the clusters's centers as the representative states in the reduced MDP [4]. However, we did not provide a theoretical justification for such a strategy. In this paper we fill this gap by showing that we can approximate KBRL's value function at any desired level of accuracy by minimizing the distance from a sampled state to the nearest representative state. Besides its theoretical interest, the bound is also relevant from a practical point of view, since it can be used in *i*KBSF to guide the on-line selection of representative states.

Finally, a third contribution of this paper is an empirical demonstration of the performance of *i*KBSF in a new, challenging control problem: the triple pole-balancing task, an extension of the well-known double pole-balancing domain. Here, *i*KBSF's ability to process a large number of transitions is crucial for achieving a high success rate, which cannot be easily replicated with batch methods.

## 2 Background

In reinforcement learning, an agent interacts with an environment in order to find a policy that maximizes the discounted sum of rewards [6]. As usual, we assume that such an interaction can be modeled as a *Markov decision process* (MDP, [7]). An MDP is a tuple $M \equiv (S, A, \mathbf{P}^a, \mathbf{r}^a, \gamma)$, where $S$ is the state space and $A$ is the (finite) action set. In this paper we are mostly concerned with MDPs with continuous state spaces, but our strategy will be to approximate such models as finite MDPs. In a finite MDP the matrix $\mathbf{P}^a \in \mathbb{R}^{|S| \times |S|}$ gives the transition probabilities associated with action $a \in A$ and the vector $\mathbf{r}^a \in \mathbb{R}^{|S|}$ stores the corresponding expected rewards. The discount factor $\gamma \in [0, 1)$ is used to give smaller weights to rewards received further in the future.

Consider an MDP $M$ with continuous state space $\mathbb{S} \subset [0, 1]^d$. Kernel-based reinforcement learning (KBRL) uses sample transitions to derive a finite MDP that approximates the continuous model [1, 2]. Let $S^a = \{(s_k^a, r_k^a, \hat{s}_k^a) | k = 1, 2, ..., n_a\}$ be sample transitions associated with action $a \in A$, where $s_k^a, \hat{s}_k^a \in \mathbb{S}$ and $r_k^a \in \mathbb{R}$. Let $\phi : \mathbb{R}^+ \mapsto \mathbb{R}^+$ be a Lipschitz continuous function and let $k_\tau(s, s')$ be a kernel function defined as $k_\tau(s, s') = \phi(\| s - s' \| / \tau)$, where $\| \cdot \|$ is a norm in $\mathbb{R}^d$ and $\tau > 0$. Finally, define the normalized kernel function associated with action $a$ as $\kappa_\tau^a(s, s_i^a) = k_\tau(s, s_i^a) / \sum_{j=1}^{n_a} k_\tau(s, s_j^a)$. The model constructed by KBRL has the following transition and reward functions:

$$\hat{P}^a(s'|s) = \begin{cases} \kappa_\tau^a(s, s_i^a), & \text{if } s' = \hat{s}_i^a, \\ 0, & \text{otherwise} \end{cases} \quad \text{and} \quad \hat{R}^a(s, s') = \begin{cases} r_i^a, & \text{if } s' = \hat{s}_i^a, \\ 0, & \text{otherwise.} \end{cases} \quad (1)$$

Since only transitions ending in the states $\hat{s}_i^a$ have a non-zero probability of occurrence, one can define a finite MDP $\hat{M}$ composed solely of these $n = \sum_a n_a$ states [2, 3]. After $\hat{V}^*$, the optimal value function of $\hat{M}$, has been computed, the value of any state-action pair can be determined as: $Q(s, a) = \sum_{i=1}^{n_a} \kappa_\tau^a(s, s_i^a) \left[ r_i^a + \gamma \hat{V}^*(\hat{s}_i^a) \right]$, where $s \in \mathbb{S}$ and $a \in A$. Ormoneit and Sen [1] proved that, if $n_a \to \infty$ for all $a \in A$ and the widths of the kernels $\tau$ shrink at an "admissible" rate, the probability of choosing a suboptimal action based on $Q(s, a)$ converges to zero.

Using dynamic programming, one can compute the optimal value function of $\hat{M}$, but the time and space required to do so grow fast with the number of states $n$ [7, 8]. Therefore, the use of KBRL leads to a dilemma: on the one hand, one wants to use as many transitions as possible to capture the dynamics of $M$, but on the other hand one wants to have an MDP $\hat{M}$ of manageable size.

Kernel-based stochastic factorization (KBSF) provides a practical way of weighing these two conflicting objectives [4]. Our algorithm compresses the information contained in KBRL's model $\hat{M}$ in an MDP $\bar{M}$ whose size is independent of the number of transitions $n$. The fundamental idea behind KBSF is the "stochastic-factorization trick", which we now summarize. Let $\mathbf{P} \in \mathbb{R}^{n \times n}$ be a transition-probability matrix and let $\mathbf{P} = \mathbf{DK}$ be a factorization in which $\mathbf{D} \in \mathbb{R}^{n \times m}$ and $\mathbf{K} \in \mathbb{R}^{m \times n}$ are stochastic matrices. Then, swapping the factors $\mathbf{D}$ and $\mathbf{K}$ yields another transition matrix $\bar{\mathbf{P}} = \mathbf{KD}$ that retains the basic topology of $\mathbf{P}$—that is, the number of recurrent classes and their respective reducibilities and periodicities [9]. The insight is that, in some cases, one can work with $\bar{\mathbf{P}}$ instead of $\mathbf{P}$; when $m \ll n$, this replacement affects significantly the memory usage and computing time.

KBSF results from the application of the stochastic-factorization trick to $\hat{M}$. Let $\bar{S} \equiv \{\bar{s}_1, \bar{s}_2, ..., \bar{s}_m\}$ be a set of representative states in $\mathbb{S}$. KBSF computes matrices $\dot{\mathbf{D}}^a \in \mathbb{R}^{n_a \times m}$ and $\dot{\mathbf{K}}^a \in \mathbb{R}^{m \times n_a}$ with elements $\dot{d}_{ij}^a = \kappa_{\bar{\tau}}(\hat{s}_i^a, \bar{s}_j)$ and $\dot{k}_{ij}^a = \kappa_{\tau}^a(\bar{s}_i, s_j^a)$, where $\kappa_{\bar{\tau}}$ is defined as $\kappa_{\bar{\tau}}(s, \bar{s}_i) = k_{\bar{\tau}}(s, \bar{s}_i)/\sum_{j=1}^m k_{\bar{\tau}}(s, \bar{s}_j)$. The basic idea of the algorithm is to replace the MDP $\hat{M}$ with $\bar{M} \equiv (\bar{S}, A, \bar{\mathbf{P}}^a, \bar{\mathbf{r}}^a, \gamma)$, where $\bar{\mathbf{P}}^a = \dot{\mathbf{K}}^a \dot{\mathbf{D}}^a$ and $\bar{\mathbf{r}}^a = \dot{\mathbf{K}}^a \mathbf{r}^a$ ($\mathbf{r}^a \in \mathbb{R}^{n_a}$ is the vector composed of sample rewards $r_i^a$). Thus, instead of solving an MDP with $n$ states, one solves a model with $m$ states only. Let $\mathbf{D}^\intercal \equiv [(\dot{\mathbf{D}}^1)^\intercal (\dot{\mathbf{D}}^2)^\intercal ... (\dot{\mathbf{D}}^{|A|})^\intercal] \in \mathbb{R}^{m \times n}$ and let $\mathbf{K} \equiv [\dot{\mathbf{K}}^1 \dot{\mathbf{K}}^2 ... \dot{\mathbf{K}}^{|A|}] \in \mathbb{R}^{m \times n}$. Based on $\bar{\mathbf{Q}}^* \in \mathbb{R}^{m \times |A|}$, the optimal action-value function of $\bar{M}$, one can obtain an approximate value function for $\hat{M}$ as $\tilde{\mathbf{v}} = \Gamma \mathbf{D} \bar{\mathbf{Q}}^*$, where $\Gamma$ is the 'max' operator applied row wise, that is, $\tilde{v}_i = \max_a (\mathbf{D}\bar{\mathbf{Q}}^*)_{ia}$. We have showed that the error in $\tilde{\mathbf{v}}$ is bounded by:

$$\|\hat{\mathbf{v}}^* - \tilde{\mathbf{v}}\|_\infty \leq \frac{1}{1-\gamma} \max_a \|\hat{\mathbf{r}}^a - \mathbf{D}\bar{\mathbf{r}}^a\|_\infty + \frac{1}{(1-\gamma)^2} \left( \bar{C} \max_i (1 - \max_j d_{ij}) + \frac{\hat{C}\gamma}{2} \max_a \|\hat{\mathbf{P}}^a - \mathbf{D}\mathbf{K}^a\|_\infty \right), \quad (2)$$

where $\|\cdot\|_\infty$ is the infinity norm, $\hat{\mathbf{v}}^* \in \mathbb{R}^n$ is the optimal value function of KBRL's MDP, $\hat{C} = \max_{a,i} \hat{r}_i^a - \min_{a,i} \hat{r}_i^a$, $\bar{C} = \max_{a,i} \bar{r}_i^a - \min_{a,i} \bar{r}_i^a$, and $\mathbf{K}^a$ is matrix $\mathbf{K}$ with all elements equal to zero except for those corresponding to matrix $\dot{\mathbf{K}}^a$ (see [4] for details).

## 3 Incremental kernel-based stochastic factorization

In the batch version of KBSF, described in Section 2, the matrices $\bar{\mathbf{P}}^a$ and vectors $\bar{\mathbf{r}}^a$ are determined using all the transitions in the corresponding sets $S^a$ simultaneously. This has two undesirable consequences. First, the construction of the MDP $\bar{M}$ requires an amount of memory of $O(n_{\max}m)$, where $n_{\max} = \max_a n_a$. Although this is a significant improvement over KBRL's memory usage, which is $O(n_{\max}^2)$, in more challenging domains even a linear dependence on $n_{\max}$ may be impractical. Second, with batch KBSF the only way to incorporate new data into the model $\bar{M}$ is to recompute the multiplication $\bar{\mathbf{P}}^a = \dot{\mathbf{K}}^a \dot{\mathbf{D}}^a$ for all actions $a$ for which there are new sample transitions available. Even if we ignore the issue of memory usage, this is clearly inefficient in terms of computation. In this section we present an incremental version of KBSF that circumvents these important limitations.

Suppose we split the set of sample transitions $S^a$ in two subsets $S_1$ and $S_2$ such that $S_1 \cap S_2 = \emptyset$ and $S_1 \cup S_2 = S^a$. Without loss of generality, suppose that the sample transitions are indexed so that $S_1 \equiv \{(s_k^a, r_k^a, \hat{s}_k^a)|k = 1, 2, ..., n_1\}$ and $S_2 \equiv \{(s_k^a, r_k^a, \hat{s}_k^a)|k = n_1 + 1, n_1 + 2, ..., n_1 + n_2 = n_a\}$. Let $\bar{\mathbf{P}}^{S_1}$ and $\bar{\mathbf{r}}^{S_1}$ be matrix $\bar{\mathbf{P}}^a$ and vector $\bar{\mathbf{r}}^a$ computed by KBSF using only the $n_1$ transitions in $S_1$ (if $n_1 = 0$, we define $\bar{\mathbf{P}}^{S_1} = \mathbf{0} \in \mathbb{R}^{m \times m}$ and $\bar{\mathbf{r}}^{S_1} = \mathbf{0} \in \mathbb{R}^m$ for all $a \in A$). We want to compute $\bar{\mathbf{P}}^{S_1 \cup S_2}$ and $\bar{\mathbf{r}}^{S_1 \cup S_2}$ from $\bar{\mathbf{P}}^{S_1}$, $\bar{\mathbf{r}}^{S_1}$, and $S_2$, without using the set of sample transition $S_1$.

We start with the transition matrices $\bar{\mathbf{P}}^a$. We know that

$$\bar{p}_{ij}^{S_1} = \sum_{t=1}^{n_1} \dot{k}_{it}^a \dot{d}_{tj}^a = \sum_{t=1}^{n_1} \frac{k_\tau(\bar{s}_i, s_t^a)}{\sum_{l=1}^{n_1} k_\tau(\bar{s}_i, s_l^a)} \frac{k_{\bar{\tau}}(\hat{s}_t^a, \bar{s}_j)}{\sum_{l=1}^m k_{\bar{\tau}}(\hat{s}_t^a, \bar{s}_l)} = \frac{1}{\sum_{l=1}^{n_1} k_\tau(\bar{s}_i, s_l^a)} \sum_{t=1}^{n_1} \frac{k_\tau(\bar{s}_i, s_t^a) k_{\bar{\tau}}(\hat{s}_t^a, \bar{s}_j)}{\sum_{l=1}^m k_{\bar{\tau}}(\hat{s}_t^a, \bar{s}_l)}.$$

To simplify the notation, define $w_i^{S_1} = \sum_{l=1}^{n_1} k_\tau(\bar{s}_i, s_l^a), w_i^{S_2} = \sum_{l=n_1+1}^{n_1+n_2} k_\tau(\bar{s}_i, s_l^a)$, and $c_{ij}^t = \frac{k_\tau(\bar{s}_i, s_t^a) k_{\bar{\tau}}(\hat{s}_t^a, \bar{s}_j)}{\sum_{l=1}^m k_{\bar{\tau}}(\hat{s}_t^a, \bar{s}_l)}$, with $t \in \{1, 2, ..., n_1 + n_2\}$. Then,

$$\bar{p}_{ij}^{S_1 \cup S_2} = \frac{1}{w_i^{S_1} + w_i^{S_2}} \left( \sum_{t=1}^{n_1} c_{ij}^t + \sum_{t=n_1+1}^{n_1+n_2} c_{ij}^t \right) = \frac{1}{w_i^{S_1} + w_i^{S_2}} \left( \bar{p}_{ij}^{S_1} w_i^{S_1} + \sum_{t=n_1+1}^{n_1+n_2} c_{ij}^t \right).$$

Now, defining $b_{ij}^{S_2} = \sum_{t=n_1+1}^{n_1+n_2} c_{ij}^t$, we have the simple update rule:

$$\boxed{\bar{p}_{ij}^{S_1 \cup S_2} = \frac{1}{w_i^{S_1} + w_i^{S_2}} \left( b_{ij}^{S_2} + \bar{p}_{ij}^{S_1} w_i^{S_1} \right)} . \tag{3}$$

We can apply similar reasoning to derive an update rule for the rewards $\bar{r}_i^a$. We know that

$$\bar{r}_i^{S_1} = \frac{1}{\sum_{l=1}^{n_1} k_\tau(\bar{s}_i, s_l^a)} \sum_{t=1}^{n_1} k_\tau(\bar{s}_i, s_t^a) r_t^a = \frac{1}{w_i^{S_1}} \sum_{t=1}^{n_1} k_\tau(\bar{s}_i, s_t^a) r_t^a .$$

Let $h_i^t = k_\tau(\bar{s}_i, s_t^a) r_t^a$, with $t \in \{1, 2, ..., n_1 + n_2\}$. Then,

$$\bar{r}_i^{S_1 \cup S_2} = \frac{1}{w_i^{S_1} + w_i^{S_2}} \left( \sum_{t=1}^{n_1} h_i^t + \sum_{t=n_1+1}^{n_1+n_2} h_i^t \right) = \frac{1}{w_i^{S_1} + w_i^{S_2}} \left( w_i^{S_1} \bar{r}_i^{S_1} + \sum_{t=n_1+1}^{n_1+n_2} h_i^t \right) .$$

Defining $e_i^{S_2} = \sum_{t=n_1+1}^{n_1+n_2} h_i^t$, we have the following update rule:

$$\boxed{\bar{r}_i^{S_1 \cup S_2} = \frac{1}{w_i^{S_1} + w_i^{S_2}} \left( e_i^{S_2} + \bar{r}_i^{S_1} w_i^{S_1} \right)} . \tag{4}$$

Since $b_{ij}^{S_2}$, $e_i^{S_2}$, and $w_i^{S_2}$ can be computed based on $S_2$ only, we can discard the sample transitions in $S_1$ after computing $\bar{\mathbf{P}}^{S_1}$ and $\bar{\mathbf{r}}^{S_1}$. To do that, we only have to keep the variables $w_i^{S_1}$. These variables can be stored in $|A|$ vectors $\mathbf{w}^a \in \mathbb{R}^m$, resulting in a modest memory overhead. Note that we can apply the ideas above recursively, further splitting the sets $S_1$ and $S_2$ in subsets of smaller size. Thus, we have a fully incremental way of computing KBSF's MDP which requires almost no extra memory.

Algorithm 1 shows a step-by-step description of how to update $\bar{M}$ based on a set of sample transitions. Using this method to update its model, KBSF's space complexity drops from $O(nm)$ to $O(m^2)$. Since the amount of memory used by KBSF is now independent of $n$, it can process an arbitrary number of sample transitions.

| **Algorithm 1** Update KBSF's MDP | **Algorithm 2** Incremental KBSF ($i$KBSF) |
|---|---|
| **Input:** $\bar{\mathbf{P}}^a, \bar{\mathbf{r}}^a, \mathbf{w}^a$ for all $a \in A$; $S^a$ for all $a \in A$ <br> **Output:** Updated $\bar{M}$ and $\mathbf{w}^a$ <br><br> **for** $a \in A$ **do** <br>    **for** $t = 1, ..., n_a$ **do** $z_t \leftarrow \sum_{l=1}^m k_{\bar{\tau}}(\hat{s}_t^a, \bar{s}_l)$ <br>    $n_a \leftarrow |S^a|$ <br>    **for** $i = 1, 2, ..., m$ **do** <br>      $w' \leftarrow \sum_{t=1}^{n_a} k_\tau(\bar{s}_i, s_t^a)$ <br>      **for** $j = 1, 2, ..., m$ **do** <br>        $b \leftarrow \sum_{t=1}^{n_a} k_\tau(\bar{s}_i, s_t^a) k_{\bar{\tau}}(\hat{s}_t^a, \bar{s}_j)/z_t$ <br>        $\bar{p}_{ij} \leftarrow \frac{1}{w_i^a + w'}(b + \bar{p}_{ij} w_i^a)$ <br>      $e \leftarrow \sum_{t=1}^{n_a} k_\tau(\bar{s}_i, s_t^a) r_t^a$ <br>      $\bar{r}_i \leftarrow \frac{1}{w_i^a + w'}(e + \bar{r}_i w_i^a)$ <br>      $w_i^a \leftarrow w_i^a + w'$ | **Input:** $\bar{s}_i$ Representative states, $i = 1, 2, ..., m$; $t_m$ Interval to update model; $t_v$ Interval to update value function; $n$ Total number of sample transitions <br> **Output:** Approximate value function $\tilde{Q}(s, a)$ <br> $\bar{\mathbf{Q}} \leftarrow$ arbitrary matrix in $\mathbb{R}^{m \times |A|}$ <br> $\bar{\mathbf{P}}^a \leftarrow \mathbf{0} \in \mathbb{R}^{m \times m}, \bar{\mathbf{r}}^a \leftarrow \mathbf{0} \in \mathbb{R}^m, \mathbf{w}^a \leftarrow \mathbf{0} \in \mathbb{R}^m, \forall a \in A$ <br> **for** $t = 1, 2, ..., n$ **do** <br>    Select $a$ based on $\tilde{Q}(s_t, a) = \sum_{i=1}^m \kappa_{\bar{\tau}}(s_t, \bar{s}_i) \bar{q}_{ia}$ <br>    Execute $a$ in $s_t$ and observe $r_t$ and $\hat{s}_t$ <br>    $S^a \leftarrow S^a \bigcup \{(s_t, r_t, \hat{s}_t)\}$ <br>    **if** ($t \mod t_m = 0$) **then** <br>      Add new representative states to $\bar{M}$ using $S^a$ <br>      Update $\bar{M}$ and $\mathbf{w}^a$ using Algorithm 1 and $S^a$ <br>      $S^a \leftarrow \emptyset$ for all $a \in A$ <br>    **if** ($t \mod t_v = 0$) update $\bar{\mathbf{Q}}$ |

Instead of assuming that $S_1$ and $S_2$ are a partition of a fixed dataset $S^a$, we can consider that $S_2$ was generated based on the policy learned by KBSF using the transitions in $S_1$. Thus, Algorithm 1 provides a flexible framework for integrating learning and planning within KBSF. A general description of the incremental version of KBSF is given in Algorithm 2. $i$KBSF updates the model $\bar{M}$ and the value function $\bar{\mathbf{Q}}$ at fixed intervals $t_m$ and $t_v$, respectively. When $t_m = t_v = n$, we recover the batch version of KBSF; when $t_m = t_v = 1$, we have an on-line method which stores no sample transitions.

Note that Algorithm 2 also allows for the inclusion of new representative states to the model $\bar{M}$. Using Algorithm 1 this is easy to do: given a new representative state $\bar{s}_{m+1}$, it suffices to set $w_{m+1}^a = 0$, $\bar{r}_{m+1}^a = 0$, and $\bar{p}_{m+1,j} = \bar{p}_{j,m+1} = 0$ for $j = 1, 2, ..., m+1$ and all $a \in A$. Then, in the following applications of Eqns (3) and (4), the dynamics of $\bar{M}$ will naturally reflect the existence of state $\bar{s}_{m+1}$.

# 4 Theoretical Results

Our previous experiments with KBSF suggest that, at least empirically, the algorithm's performance improves as $m \to n$ [4] . In this section we present theoretical results that confirm this property. The results below are particularly useful for *i*KBSF because they provide practical guidance towards where and when to add new representative states.

Suppose we have a fixed set of sample transitions $S^a$. We will show that, if we are free to define the representative states, then we can use KBSF to approximate KBRL's solution to any desired level of accuracy. To be more precise, let $\mathfrak{d}^* \equiv \max_{a,i} \min_j \| \hat{s}_i^a - \bar{s}_j \|$, that is, $\mathfrak{d}^*$ is the maximum distance from a sampled state $\hat{s}_i^a$ to the closest representative state. We will show that, by minimizing $\mathfrak{d}^*$, we can make $\| \hat{\mathbf{v}}^* - \tilde{\mathbf{v}} \|_\infty$ as small as desired (*cf.* Eqn (2)).

Let $\hat{s}_*^a \equiv \hat{s}_k^a$ with $k = \text{argmax}_i \min_j \| \hat{s}_i^a - \bar{s}_j \|$ and $\bar{s}_*^a \equiv \bar{s}_h$ where $h = \text{argmin}_j \| \hat{s}_*^a - \bar{s}_j \|$, that is, $\hat{s}_*^a$ is the sampled state in $S^a$ whose distance to the closest representative state is maximal, and $\bar{s}_*^a$ is the representative state that is closest to $\hat{s}_*^a$. Using these definitions, we can select the pair $(\hat{s}_*^a, \bar{s}_*^a)$ that maximizes $\| \hat{s}_*^a - \bar{s}_*^a \|$: $\hat{s}_* \equiv \hat{s}_*^b$ and $\bar{s}_* \equiv \bar{s}_*^b$ where $b = \text{argmax}_a \| \hat{s}_*^a - \bar{s}_*^a \|$. Obviously, $\| \hat{s}_* - \bar{s}_* \| = \mathfrak{d}^*$.

We make the following simple assumptions: (i) $\hat{s}_*^a$ and $\bar{s}_*^a$ are unique for all $a \in A$, (ii) $\int_0^\infty \phi(x)dx \le L_\phi < \infty$, (iii) $\phi(x) \ge \phi(y)$ if $x < y$, (iv) $\exists A_\phi, \lambda_\phi > 0, \exists B_\phi \ge 0$ such that $A_\phi \exp(-x) \le \phi(x) \le \lambda_\phi A_\phi \exp(-x)$ if $x \ge B_\phi$. Assumption (iv) implies that the kernel function $\phi$ will eventually decay exponentially. We start by introducing the following definition:

**Definition 1.** *Given $\alpha \in (0, 1]$ and $s, s' \in \mathbb{S}$, the $\alpha$-radius of $k_\tau$ with respect to $s$ and $s'$ is defined as* $\rho(k_\tau, s, s', \alpha) = \max\{x \in \mathbb{R}^+ | \phi(x/\tau) = \alpha k_\tau(s, s')\}$.

The existence of $\rho(k_\tau, s, s', \alpha)$ is guaranteed by assumptions (ii) and (iii) and the fact that $\phi$ is continuous [1]. To provide some intuition on the meaning of the $\alpha$-radius of $k_\tau$, suppose that $\phi$ is strictly decreasing and let $c = \phi(\| s - s' \| / \tau)$. Then, there is a $s'' \in \mathbb{S}$ such that $\phi(\| s - s'' \| / \tau) = \alpha c$. The radius of $k_\tau$ in this case is $\| s - s'' \|$. It should be thus obvious that $\rho(k_\tau, s, s', \alpha) \ge \| s - s' \|$. We can show that $\rho$ has the following properties (proved in the supplementary material):

**Property 1.** *If $\| s - s' \| < \| s - s'' \|$, then $\rho(k_\tau, s, s', \alpha) \le \rho(k_\tau, s, s'', \alpha)$.*

**Property 2.** *If $\alpha < \alpha'$, then $\rho(k_\tau, s, s', \alpha) > \rho(k_\tau, s, s', \alpha')$.*

**Property 3.** *For $\alpha \in (0, 1)$ and $\varepsilon > 0$, there is a $\delta > 0$ such that $\rho(k_\tau, s, s', \alpha) - \| s - s' \| < \varepsilon$ if $\tau < \delta$.*

We now introduce a notion of dissimilarity between two states $s, s' \in \mathbb{S}$ which is induced by a specific set of sample transitions $S^a$ and the choice of kernel function:

**Definition 2.** *Given $\beta > 0$, the $\beta$-dissimilarity between $s$ and $s'$ with respect to $\kappa_\tau^a$ is defined as*

$$\psi(\kappa_\tau^a, s, s', \beta) = \begin{cases} \sum_{k=1}^{n_a} |\kappa_\tau^a(s, s_k^a) - \kappa_\tau^a(s', s_k^a)|, & \text{if } \| s - s' \| \le \beta, \\ 0, & \text{otherwise.} \end{cases}$$

The parameter $\beta$ defines the volume of the ball within which we want to compare states. As we will see, this parameter links Definitions 1 and 2. Note that $\psi(\kappa_\tau^a, s, s', \beta) \in [0, 2]$. It is possible to show that $\psi$ satisfies the following property (see supplementary material):

**Property 4.** *For $\beta > 0$ and $\varepsilon > 0$, there is a $\delta > 0$ such that $\psi(\kappa_\tau^a, s, s', \beta) < \varepsilon$ if $\| s - s' \| < \delta$.*

Definitions 1 and 2 allow us to enunciate the following result:

**Lemma 1.** *For any $\alpha \in (0, 1]$ and any $t \ge m - 1$, let $\rho^a = \rho(k_{\bar{\tau}}, \hat{s}_*^a, \bar{s}_*^a, \alpha/t)$, let $\psi_\rho^a = \max_{i,j} \psi(\kappa_\tau^a, \hat{s}_i^a, \bar{s}_j, \rho^a)$, and let $\psi_{\max}^a = \max_{i,j} \psi(\kappa_\tau^a, \hat{s}_i^a, \bar{s}_j, \infty)$. Then,*

$$\| \mathbf{P}^a - \mathbf{D}\mathbf{K}^a \|_\infty \le \frac{1}{1 + \alpha} \psi_\rho^a + \frac{\alpha}{1 + \alpha} \psi_{\max}^a. \tag{5}$$

*Proof.* See supplementary material.

Since $\psi_{\max}^a \ge \psi_\rho^a$, one might think at first that the right-hand side of Eqn (5) decreases monotonically as $\alpha \to 0$. This is not necessarily true, though, because $\psi_\rho^a \to \psi_{\max}^a$ as $\alpha \to 0$ (see Property 2). We are finally ready to prove the main result of this section.

**Proposition 1.** *For any $\varepsilon > 0$, there are $\delta_1, \delta_2 > 0$ such that $\|\hat{\mathbf{v}}^* - \tilde{\mathbf{v}}\|_\infty < \varepsilon$ if $\mathfrak{d}^* < \delta_1$ and $\bar{\tau} < \delta_2$.*

*Proof.* Let $\check{\mathbf{r}} \equiv [(\mathbf{r}^1)^\mathsf{T}, (\mathbf{r}^2)^\mathsf{T}, ..., (\mathbf{r}^{|A|})^\mathsf{T}]^\mathsf{T} \in \mathbb{R}^n$. From Eqn (1) and the definition of $\bar{\mathbf{r}}^a$, we can write

$$\|\hat{\mathbf{r}}^a - \mathbf{D}\bar{\mathbf{r}}^a\|_\infty = \|\hat{\mathbf{P}}^a\check{\mathbf{r}} - \mathbf{D}\dot{\mathbf{K}}^a\mathbf{r}^a\|_\infty = \|\hat{\mathbf{P}}^a\check{\mathbf{r}} - \mathbf{D}\mathbf{K}^a\check{\mathbf{r}}\|_\infty = \|(\hat{\mathbf{P}}^a - \mathbf{D}\mathbf{K}^a)\check{\mathbf{r}}\|_\infty \leq \|\hat{\mathbf{P}}^a - \mathbf{D}\mathbf{K}^a\|_\infty \|\check{\mathbf{r}}\|_\infty. \quad (6)$$

Thus, plugging Eqn (6) back into Eqn (2), it is clear that there is a $\eta > 0$ such that $\|\hat{\mathbf{v}}^* - \tilde{\mathbf{v}}\|_\infty < \varepsilon$ if $\max_a \|\hat{\mathbf{P}}^a - \mathbf{D}\mathbf{K}^a\|_\infty < \eta$ and $\max_i (1 - \max_j d_{ij}) < \eta$. We start by showing that if $\mathfrak{d}^*$ and $\bar{\tau}$ are small enough, then $\max_a \|\hat{\mathbf{P}}^a - \mathbf{D}\mathbf{K}^a\|_\infty < \eta$. From Lemma 1 we know that, for any set of $m \leq n$ representative states, and for any $\alpha \in (0, 1]$, the following must hold:

$$\max_a \|\mathbf{P}^a - \mathbf{D}\mathbf{K}^a\|_\infty \leq (1+\alpha)^{-1}\psi_\rho + \alpha(1+\alpha)^{-1}\psi_{\text{MAX}},$$

where $\psi_{\text{MAX}} = \max_{a,i,s} \psi(\mathbf{k}_\tau, \hat{s}_i^a, s, \infty)$ and $\psi_\rho = \max_a \psi_\rho^a = \max_{a,i,j} \psi(\kappa_\tau^a, \hat{s}_i^a, \bar{s}_j, \rho^a)$, with $\rho^a = \rho(\mathbf{k}_{\bar{\tau}}, \hat{s}_*^a, \bar{s}_*^a, \alpha/(n-1))$. Note that $\psi_{\text{MAX}}$ is independent of the representative states. Define $\alpha$ such that $\alpha/(1+\alpha)\psi_{\text{MAX}} < \eta$. We have to show that, if we define the representative states in such a way that $\mathfrak{d}^*$ is small enough, and set $\bar{\tau}$ accordingly, then we can make $\psi_\rho < (1-\alpha)\eta - \alpha\psi_{\text{MAX}} \equiv \eta'$. From Property 4 we know that there is a $\delta_1 > 0$ such that $\psi_\rho < \eta'$ if $\rho^a < \delta_1$ for all $a \in A$. From Property 1 we know that $\rho^a \leq \rho(\mathbf{k}_{\bar{\tau}}, \hat{s}_*, \bar{s}_*, \alpha/(n-1))$ for all $a \in A$. From Property 3 we know that, for any $\varepsilon' > 0$, there is a $\delta' > 0$ such that $\rho(\mathbf{k}_{\bar{\tau}}, \hat{s}_*, \bar{s}_*, \alpha/(n-1)) < \mathfrak{d}^* + \varepsilon'$ if $\bar{\tau} < \delta'$. Therefore, if $\mathfrak{d}^* < \delta_1$, we can take any $\varepsilon' < \delta_1 - \mathfrak{d}^*$ to have an upper bound $\delta'$ for $\bar{\tau}$. It remains to show that there is a $\delta > 0$ such that $\min_i \max_j d_{ij} > 1 - \eta$ if $\bar{\tau} < \delta$. Recalling that $d_{ij}^a = \mathbf{k}_{\bar{\tau}}(\hat{s}_i^a, \bar{s}_j)/\sum_{k=1}^m \mathbf{k}_{\bar{\tau}}(\hat{s}_i^a, \bar{s}_k)$, let $h = \arg\max_j \mathbf{k}_{\bar{\tau}}(\hat{s}_i^a, \bar{s}_j)$, and let $y_i^a = \mathbf{k}_{\bar{\tau}}(\hat{s}_i^a, \bar{s}_h)$ and $\check{y}_i^a = \max_{j \neq h} \mathbf{k}_{\bar{\tau}}(\hat{s}_i^a, \bar{s}_j)$. Then, for any $i$, $\max_j d_{ij}^a = y_i^a / \left(y_i^a + \sum_{j \neq h} \mathbf{k}_{\bar{\tau}}(\hat{s}_i^a, \bar{s}_j)\right) \geq y_i^a/(y_i^a + (m-1)\check{y}_i^a)$. From Assump. (i) and Prop. 3 we know that there is a $\delta_i^a > 0$ such that $y_i^a > (m-1)(1-\eta)\check{y}_i^a/\eta$ if $\bar{\tau} < \delta_i^a$. Thus, by making $\delta = \min_{a,i} \delta_i^a$, we can guarantee that $\min_i \max_j d_{ij} > 1 - \eta$. If we take $\delta_2 = \min(\delta, \delta')$, the result follows. $\quad\square$

Proposition 1 tells us that, regardless of the specific reinforcement-learning problem at hand, if the distances between sampled states and the respective nearest representative states are small enough, then we can make KBSF's approximation of KBRL's value function as accurate as desired by setting $\bar{\tau}$ to a small value. How small $\mathfrak{d}^*$ and $\bar{\tau}$ should be depends on the particular choice of kernel $\mathbf{k}_\tau$ and on the characteristics of the sets of transitions $S^a$. Of course, a fixed number $m$ of representative states imposes a minimum possible value for $\mathfrak{d}^*$, and if this value is not small enough decreasing $\bar{\tau}$ may actually hurt the approximation. Again, the optimal value for $\bar{\tau}$ in this case is problem-dependent.

Our result supports the use of a local approximation based on representative states spread over the state space $\mathbb{S}$. This is in line with the quantization strategies used in batch-mode kernel-based reinforcement learning to define the states $\bar{s}_j$ [4, 5]. In the case of on-line learning, we have to adaptively define the representative states $\bar{s}_j$ as the sample transitions come in. One can think of several ways of doing so [10]. In the next section we show a simple strategy for adding representative states which is based on the theoretical results presented in this section.

## 5 Empirical Results

We now investigate the empirical performance of the incremental version of KBSF. We start with a simple task in which *i*KBSF is contrasted with batch KBSF. Next we exploit the scalability of *i*KBSF to solve a difficult control task that, to the best of our knowledge, has never been solved before.

We use the "puddle world" problem as a proof of concept [11]. In this first experiment we show that *i*KBSF is able to recover the model that would be computed by its batch counterpart. In order to do so, we applied Algorithm 2 to the puddle-world task using a random policy to select actions. Figure 1a shows the result of such an experiment when we vary the parameters $t_m$ and $t_v$. Note that the case in which $t_m = t_v = 8000$ corresponds to the batch version of KBSF. As expected, the performance of KBSF decision policies improves gradually as the algorithm goes through more sample transitions, and in general the intensity of the improvement is proportional to the amount of data processed. More important, the performance of the decision policies after all sample transitions have been processed is essentially the same for all values of $t_m$ and $t_v$, which shows that *i*KBSF can be used as a tool to circumvent KBSF's memory demand (which is linear in $n$). Thus, if one has a batch of sample transitions that does not fit in the available memory, it is possible to split the data in chunks of smaller sizes and still get the same value-function approximation that would

be computed if the entire data set were processed at once. As shown in Figure 1b, there is only a small computational overhead associated with such a strategy (this results from unnormalizing and normalizing the elements of $\bar{\mathbf{P}}^a$ and $\bar{\mathbf{r}}^a$ several times through update rules (3) and (4)).

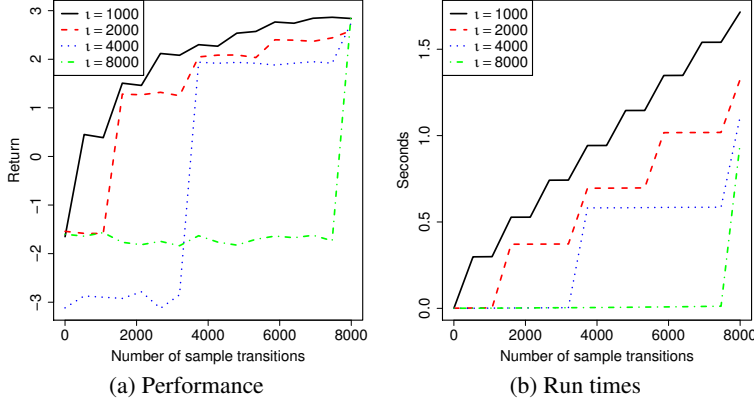

(a) Performance                 (b) Run times

Figure 1: Results on the puddle-world task averaged over 50 runs. *i*KBSF used 100 representative states evenly distributed over the state space and $t_m = t_v = \iota$ (see legends). Sample transitions were collected by a random policy. The agents were tested on two sets of states surrounding the "puddles": a $3 \times 3$ grid over $[0.1, 0.3] \times [0.3, 0.5]$ and the four states $\{0.1, 0.3\} \times \{0.9, 1.0\}$.

But *i*KBSF is more than just a tool for avoiding the memory limitations associated with batch learning. We illustrate this fact with a more challenging RL task. Pole balancing has a long history as a benchmark problem because it represents a rich class of unstable systems [12, 13, 14]. The objective in this task is to apply forces to a wheeled cart moving along a limited track in order to keep one or more poles hinged to the cart from falling over [15]. There are several variations of the problem with different levels of difficulty; among them, balancing two poles at the same time is particularly hard [16]. In this paper we raise the bar, and add a third pole to the pole-balancing task. We performed our simulations using the parameters usually adopted with the double pole task, except that we added a third pole with the same length and mass as the longer pole [15]. This results in a problem with an 8-dimensional state space $\mathbb{S}$.

In our experiments with the double-pole task, we used 200 representative states and $10^6$ sample transitions collected by a random policy [4]. Here we start our experiment with triple pole-balancing using exactly the same configuration, and then we let KBSF refine its model $\bar{M}$ by incorporating more sample transitions through update rules (3) and (4). Specifically, we used Algorithm 2 with a 0.3-greedy policy, $t_m = t_v = 10^6$, and $n = 10^7$. Policy iteration was used to compute $\bar{\mathbf{Q}}^*$ at each value-function update. As for the kernels, we adopted Gaussian functions with widths $\tau = 100$ and $\bar{\tau} = 1$ (to improve efficiency, we used a KD-tree to only compute the 50 largest values of $k_\tau(\bar{s}_i, \cdot)$ and the 10 largest values of $k_{\bar{\tau}}(\hat{s}_i^a, \cdot)$). Representative states were added to the model on-line every time the agent encountered a sample state $\hat{s}_i^a$ for which $k_{\bar{\tau}}(\hat{s}_i^a, \bar{s}_j) < 0.01$ for all $j \in 1, 2, ..., m$ (this corresponds to setting the maximum allowed distance $\mathfrak{d}^*$ from a sampled state to the closest representative state).

We compare *i*KBSF with fitted *Q*-iteration using an ensemble of 30 trees generated by Ernst et al.'s extra-trees algorithm [17]. We chose this algorithm because it has shown excellent performance in both benchmark and real-world reinforcement-learning tasks [17, 18].[1] Since this is a batch-mode learning method, we used its result on the initial set of $10^6$ sample transitions as a baseline for our empirical evaluation. To build the trees, the number of cut-directions evaluated at each node was fixed at $\dim(\mathbb{S}) = 8$, and the minimum number of elements required to split a node, denoted here by $\eta_{\min}$, was first set to 1000 and then to 100. The algorithm was run for 50 iterations, with the structure of the trees fixed after the 10th iteration.

As shown in Figure 2a, both fitted *Q*-iteration and batch KBSF perform poorly in the triple pole-balancing task, with average success rates below 55%. This suggests that the amount of data used

by these algorithms is insufficient to describe the dynamics of the control task. Of course, we could give more sample transitions to fitted $Q$-iteration and batch KBSF. Note however that, since they are batch-learning methods, there is an inherent limit on the amount of data that these algorithms can use to construct their approximation. In contrast, the amount of memory required by $i$KBSF is independent of the number of sample transitions $n$. This fact together with the fact that KBSF's computational complexity is only linear in $n$ allow our algorithm to process a large amount of data within a reasonable time. This can be observed in Figure 2b, which shows that $i$KBSF can build an approximation using $10^7$ transitions in under 20 minutes. As a reference for comparison, fitted $Q$-iteration using $\eta_{min} = 1000$ took an average of 1 hour and 18 minutes to process 10 times less data.

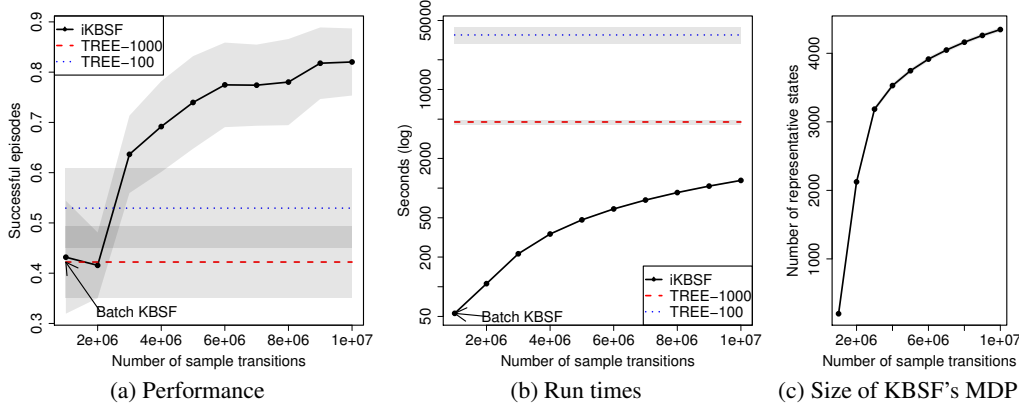

| (a) Performance | (b) Run times | (c) Size of KBSF's MDP |

Figure 2: Results on the triple pole-balancing task averaged over 50 runs. The values correspond to the fraction of episodes initiated from the test states in which the 3 poles could be balanced for 3000 steps (one minute of simulated time). The test set was composed of 256 states equally distributed over the hypercube defined by $\pm[1.2\,\mathrm{m}, 0.24\,\mathrm{m/s}, 18^o, 75^o/\mathrm{s}, 18^o, 150^o/\mathrm{s}, 18^o, 75^o/\mathrm{s}]$. Shadowed regions represent 99% confidence intervals.

As shown in Figure 2a, the ability of $i$KBSF to process a large number of sample transitions allows our algorithm to achieve a success rate of approximately 80%. This is similar to the performance of batch KBSF on the double-pole version of the problem [4]. The good performance of $i$KBSF on the triple pole-balancing task is especially impressive when we recall that the decision policies were evaluated on a set of test states representing all possible directions of inclination of the three poles. In order to achieve the same level of performance with KBSF, approximately 2 Gb of memory would be necessary, even using sparse kernels, whereas $i$KBSF used less than 0.03 Gb of memory.

To conclude, observe in Figure 2c how the number of representative states $m$ grows as a function of the number of sample transitions processed by KBSF. As expected, in the beginning of the learning process $m$ grows fast, reflecting the fact that some relevant regions of the state space have not been visited yet. As more and more data come in, the number of representative states starts to stabilize.

## 6 Conclusion

This paper presented two contributions, one practical and one theoretical. The practical contribution is $i$KBSF, the incremental version of KBSF. $i$KBSF retains all the nice properties of its precursor: it is simple, fast, and enjoys good theoretical guarantees. However, since its memory complexity is independent of the number of sample transitions, $i$KBSF can be applied to datasets of any size, and it can also be used on-line. To show how $i$KBSF's ability to process large amounts of data can be useful in practice, we used the proposed algorithm to learn how to simultaneously balance three poles, a difficult control task that had never been solved before.

As for the theoretical contribution, we showed that KBSF can approximate KBRL's value function at any level of accuracy by minimizing the distance between sampled states and the closest representative state. This supports the quantization strategies usually adopted in kernel-based RL, and also offers guidance towards where and when to add new representative states in on-line learning.

**Acknowledgments** The authors would like to thank Amir massoud Farahmand for helpful discussions regarding this work. Funding for this research was provided by the National Institutes of Health (grant R21 DA019800) and the NSERC Discovery Grant program.

## Footnotes

[1]Another reason for choosing fitted *Q*-iteration was that some of the most natural competitors of *i*KBSF have already been tested on the simpler double pole-balancing task, with disappointing results [19, 4].

# References

[1] D. Ormoneit and S. Sen. Kernel-based reinforcement learning. *Machine Learning*, 49 (2–3): 161–178, 2002.

[2] D. Ormoneit and P. Glynn. Kernel-based reinforcement learning in average-cost problems. *IEEE Transactions on Automatic Control*, 47(10):1624–1636, 2002.

[3] N. Jong and P. Stone. Kernel-based models for reinforcement learning in continuous state spaces. In *Proceedings of the International Conference on Machine Learning (ICML)—Workshop on Kernel Machines and Reinforcement Learning*, 2006.

[4] A. M. S. Barreto, D. Precup, and J. Pineau. Reinforcement learning using kernel-based stochastic factorization. In *Advances in Neural Information Processing Systems (NIPS)*, pages 720–728, 2011.

[5] B. Kveton and G. Theocharous. Kernel-based reinforcement learning on representative states. In *Proceedings of the AAAI Conference on Artificial Intelligence (AAAI)*, pages 124–131, 2012.

[6] R. S. Sutton and A. G. Barto. *Reinforcement Learning: An Introduction*. MIT Press, 1998.

[7] M. L. Puterman. *Markov Decision Processes—Discrete Stochastic Dynamic Programming*. John Wiley & Sons, Inc., 1994.

[8] M. L. Littman, T. L. Dean, and L. P. Kaelbling. On the complexity of solving Markov decision problems. In *Proceedings of the Conference on Uncertainty in Artificial Intelligence* (UAI), pages 394–402, 1995.

[9] A. M. S. Barreto and M. D. Fragoso. Computing the stationary distribution of a finite Markov chain through stochastic factorization. *SIAM Journal on Matrix Analysis and Applications*, 32: 1513–1523, 2011.

[10] Y. Engel, S. Mannor, and R. Meir. The kernel recursive least squares algorithm. *IEEE Transactions on Signal Processing*, 52:2275–2285, 2003.

[11] R. S. Sutton. Generalization in reinforcement learning: Successful examples using sparse coarse coding. In *Advances in Neural Information Processing Systems* (NIPS), pages 1038–1044, 1996.

[12] D. Michie and R. Chambers. BOXES: An experiment in adaptive control. *Machine Intelligence 2*, pages 125–133, 1968.

[13] C. W. Anderson. *Learning and Problem Solving with Multilayer Connectionist Systems*. PhD thesis, Computer and Information Science, University of Massachusetts, 1986.

[14] A. G. Barto, R. S. Sutton, and C. W. Anderson. Neuronlike adaptive elements that can solve difficult learning control problems. *IEEE Transactions on Systems, Man, and Cybernetics*, 13: 834–846, 1983.

[15] F. J. Gomez. *Robust non-linear control through neuroevolution*. PhD thesis, The University of Texas at Austin, 2003.

[16] A. P. Wieland. Evolving neural network controllers for unstable systems. In *Proceedings of the International Joint Conference on Neural Networks (IJCNN)*, pages 667–673, 1991.

[17] D. Ernst, P. Geurts, and L. Wehenkel. Tree-based batch mode reinforcement learning. *Journal of Machine Learning Research*, 6:503–556, 2005.

[18] D. Ernst, G. B. Stan, J. Gonçalves, and L. Wehenkel. Clinical data based optimal STI strategies for HIV: a reinforcement learning approach. In *Proceedings of the IEEE Conference on Decision and Control* (CDC), pages 124–131, 2006.

[19] F. Gomez, J. Schmidhuber, and R. Miikkulainen. Efficient non-linear control through neuroevolution. In *Proceedings of the European Conference on Machine Learning* (ECML), pages 654–662, 2006.

